# Know Thy Neighbour:
# A Normative Theory of Synaptic Depression

**Jean-Pascal Pfister**

Computational & Biological Learning Lab
Department of Engineering, University of Cambridge
Trumpington Street, Cambridge CB2 1PZ, United Kingdom
jean-pascal.pfister@eng.cam.ac.uk

**Peter Dayan**

Gatsby Computational Neuroscience Unit, UCL
17 Queen Square, London WC1N 3AR, United Kingdom
dayan@gatsby.ucl.ac.uk

**Máté Lengyel**

Computational & Biological Learning Lab
Department of Engineering, University of Cambridge
Trumpington Street, Cambridge CB2 1PZ, United Kingdom
m.lengyel@eng.cam.ac.uk

## Abstract

Synapses exhibit an extraordinary degree of short-term malleability, with release probabilities and effective synaptic strengths changing markedly over multiple timescales. From the perspective of a fixed computational operation in a network, this seems like a most unacceptable degree of added variability. We suggest an alternative theory according to which short-term synaptic plasticity plays a normatively-justifiable role. This theory starts from the commonplace observation that the spiking of a neuron is an incomplete, digital, report of the analog quantity that contains all the critical information, namely its membrane potential. We suggest that a synapse solves the inverse problem of estimating the pre-synaptic membrane potential from the spikes it receives, acting as a recursive filter. We show that the dynamics of short-term synaptic depression closely resemble those required for optimal filtering, and that they indeed support high quality estimation. Under this account, the local postsynaptic potential and the level of synaptic resources track the (scaled) mean and variance of the estimated presynaptic membrane potential. We make experimentally testable predictions for how the statistics of subthreshold membrane potential fluctuations and the form of spiking non-linearity should be related to the properties of short-term plasticity in any particular cell type.

## 1   Introduction

Far from being static relays, synapses are complex dynamical elements. The effect of a spike from a presynaptic neuron on its postsynaptic partner depends on the history of the activity of both pre- and postsynaptic neurons, and thus the efficacy of a synapse undergoes perpetual modification. These changes in efficacy can last from hundreds of milliseconds or minutes (short-term plasticity) to hours or months (long-term plasticity). Short-term plasticity typically only depends on the firing pattern

of the presynaptic cell [1]; short term depression gradually diminishes the postsynaptic effects of presynaptic spikes that arrive in quick succession (Fig. 1A). Given the prominence and ubiquity of synaptic depression in cortical (and subcortical) synapses [2], it is pressing to identify its computational role(s).

There have thus been various important suggestions for the functional significance of synaptic depression, including – just to name a few – low-pass filtering of inputs [3], rendering postsynaptic responses insensitive to the absolute intensity of presynaptic activity [4, 5], and decorrelating input spike sequences [6]. However, important though they must be for select neural systems, these suggestions have a piecemeal flavor – for instance, chaining together stages of low-pass filtering would lead to trivial responding.

Here, we propose a theory according which synaptic depression solves a computational problem that is faced by any neural population in which neurons represent and compute with analog quantities, but communicate with discrete spikes. For convenience, we assume this analog quantity to be the membrane potential, but, via a non-linear transformation [7], it could equally well be an analog firing rate. That is, we assume that network computations require the evolution of the membrane potential of a neuron to be a function of the membrane potentials of its presynaptic partners. However, such a neuron does not have (at least not directly, see [8] for an example of indirect interaction) access to these membrane potentials, but rather only to the spikes to which they lead, and so it faces a key estimation problem.

Thus, much as in the vein of standard textbook presentations, the operation of a neuron can be logically broken down into three concurrent processes, each running in its dedicated functional compartment: 1) the neuron's afferent synapses (e.g. spines) estimate the membrane potential of its presynaptic partners, scaled according to the rules of the network computation; 2) the neuron's soma-dendritic compartment follows the membrane potential-dependent dynamics and post-synaptic integration also determined by the computation; and 3) its axon generates action potentials that are broadcasted to its efferent synapses (and possibly back to the other compartments, eg. for long-term plasticity). It is in the indispensable first estimation step that we suggest synaptic depression to be involved.

In Section 2 we formalise the problem of estimating presynaptic membrane potentials as an instance of Bayesian inference, and derive an online recursive estimator for it. Given suitable assumptions about presynaptic membrane potential dynamics and spike generation, this optimal estimator can be written in closed form exactly [9, 10]. In Section 3, we introduce a canonical model of postsynaptic membrane potential and synaptic depression dynamics, and show how it relates to the optimal estimator derived earlier. In Section 4, we present results from numerical simulations showing the quality with which synaptic depression can approximate the performance of the optimal estimator, and how much is gained relative to a static synapse without synaptic depression. Finally, in Section 5, we sum up, suggest experimentally testable predictions, and discuss possible extensions of this work, eg. to incorporate other forms of short-term synaptic plasticity.

## 2 Bayesian estimation of presynaptic membrane potentials

The Bayesian estimation problem that needs to be solved by a synapse involves inferring the posterior distribution $p(u_t|s_{1..t})$ over the presynaptic membrane potential $u_t$ at time step $t$ (for discretized time), given the spikes seen from the presynaptic cell up to that time step, $s_{1..t}$. We first define a statistical (generative) model of presynaptic membrane potential fluctuations and spiking, and then derive the estimator that is appropriate for it.

The generative model involves two simplifying assumptions (Fig. 1B). First we assume that presynaptic membrane potential dynamics are Markovian

$$p(u_t|u_{1..t-1}) = p(u_t|u_{t-1}) \tag{1}$$

In particular, we assume that the presynaptic membrane potential evolves as an Ornstein-Uhlenbeck (OU) process, given (again, in discretized time) by

$$u_t = u_{t-1} - \theta(u_{t-1} - u_{\mathrm{r}})\Delta t + W_t\sqrt{\Delta t}, \qquad W_t \overset{\mathrm{iid}}{\sim} \mathcal{N}(W_t; 0, \sigma_{\mathrm{W}}^2) \tag{2}$$

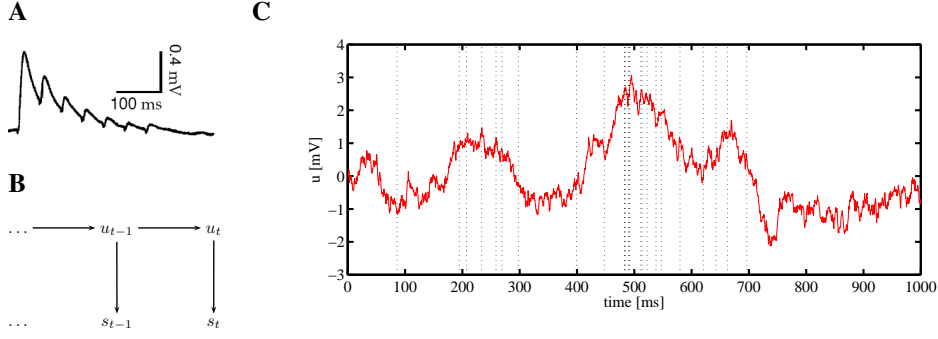

Figure 1: **A.** Synaptic depression: postsynaptic responses to a train of presynaptic action potentials (not shown) at $40\,\mathrm{Hz}$. (Reproduced from [11], adapted from [12].) **B.** Graphical model of the process generating presynaptic subthreshold membrane potential fluctuations, $u$, and spikes, $s$. The membrane potential evolves according to a first-order Markov process, the Ornstein-Uhlenbeck (OU) process (Eqs. 1-2). The probability of generating a spike at time $t$ ($s_t = 1$) depends only on the current membrane potential, $u_t$, and is determined by a non-linear Poisson (NP) model (Eqs. 3-5). **C.** Sample membrane potential trace (*red line*) and spike timings (*vertical black dotted lines*) generated by the OU-NP process; with $u_{\mathrm{r}} = 0\,\mathrm{mV}$, $\theta^{-1} = 100\,\mathrm{ms}$, $\sigma_{\mathrm{W}}^2 = 0.02\,\mathrm{mV}^2/\mathrm{ms} \to \sigma_{\mathrm{OU}}^2 = 1\,\mathrm{mV}^2$, $\beta^{-1} = 1\,\mathrm{mV}$, and $g_0 = 10\,\mathrm{Hz}$.

where $1/\theta$ is the time constant with which the membrane potential decays back to its resting value, $u_{\mathrm{r}}$, and $\Delta t$ is the size of the discretized time bins. Because both $\theta$ and $\sigma_{\mathrm{W}}$ are assumed to be constant, the variance of the presynaptic membrane potential, $\sigma_{\mathrm{OU}}^2 = \sigma_{\mathrm{W}}^2/2\theta$, is stationary.

The second assumption is that spiking activity at any time only depends on the membrane potential at that time:

$$p(s_t|u_{1..t}) = p(s_t|u_t) \tag{3}$$

In particular, we assume that the spike generating mechanism is an inhomogeneous Poisson process (Fig. 1C). Thus, at time step $t$, the neuron emits a spike ($s_t = 1$) with probability $g(u_t)\Delta t$, and therefore the spiking probability $p(s_t|u_t)$ given the membrane potential can be written as:

$$p(s_t|u_t) = [g(u_t)\Delta t]^{s_t} \, [1 - g(u)\Delta t]^{(1-s_t)} \tag{4}$$

We further assume that the transfer function, $g(u)$, is exponential[1]:

$$g(u) = g_0 \exp(\beta u) \tag{5}$$

where $\beta$ determines the stochasticity of spiking. In the limit $\beta \to \infty$ the spiking process is deterministic, i.e. if the membrane potential, $u$, is bigger than zero, the neuron emits a spike, and if $u < 0$, the neuron does not fire.

Estimating on-line the membrane potential of the presynaptic cell from its spiking history amounts to computing the posterior probability distribution, $p(u_t|s_{1..t})$. Since equations 1 and 3 define a hidden Markov model, the posterior can be written in a recursive form:

$$p(u_t|s_{1..t}) \propto p(s_t|u_t) \int p(u_t|u_{t-1}) \, p(u_{t-1}|s_{1..t-1}) \, du_{t-1} \tag{6}$$

That is, the posterior at time step $t$, $p(u_t|s_{1..t})$, can be computed by combining information from the current time step with the posterior obtained at the previous time step, $p(u_{t-1}|s_{1..t-1})$. Note that even though inference can be performed recursively, and the hidden dynamics is linear-Gaussian (Eq. 2), the (extended) Kalman filter cannot be used here for inference because the measurement does not involve additive Gaussian noise, but rather comes from the stochasticity of the spiking process (Eqs. 4-5).

Performing recursive inference (filtering), as described by equation 6, under the generative model described by equations 1-5 results in a posterior distribution that is Gaussian, $u_t|s_{1..t} \sim \mathcal{N}(u_t; \mu, \sigma^2)$ (see Supplementary Information). The mean and variance of this Gaussian evolve (in continuous time, by taking the limit $\Delta t \to 0$) as:

$$\dot{\mu} = -\theta(\mu - u_r) + \beta\sigma^2(S(t) - \gamma) \tag{7}$$

$$\dot{\sigma}^2 = -2\theta\left(\sigma^2 - \sigma_{OU}^2\right) - \gamma\beta^2\sigma^4 \tag{8}$$

with the normalisation factor given by

$$\gamma = \langle g_0 \exp(\beta u)\rangle_{u_t|s_{1..t}} = g_0 \exp\left(\beta\mu + \frac{\beta^2\sigma^2}{2}\right) \tag{9}$$

where $S(t)$ is the spike train of the presynaptic cell (represented as a sum of Dirac delta functions). (A similar, but not identical, derivation can be found in [9]).

Equation 7 indicates that each time a spike is observed, the estimated membrane potential should increase proportionally to the uncertainty (variance) about the current estimate. This estimation uncertainty then decreases each time a spike is observed (Eqs. 8-9). As Fig. 2A shows, the higher the presynaptic membrane potential is, the more spikes are emitted (because the instantaneous firing rate is a monotonic function of membrane potential, see Eq. 5), and therefore the smaller the posterior variance becomes. Therefore the estimation error is smaller for higher membrane potential (see Fig. 2B). Conversely, in the absence of spikes, the estimated membrane potential decreases while the variance increases back to its asymptotic value. Fig. 2C shows that the representation of uncertainty about the membrane potential by $\sigma^2$ is self-consistent because it is predictive of the error of the mean estimator, $\mu$.

The first term on the r.h.s of equation 7 comes from the prior knowledge about the membrane potential dynamics. The second term comes from the likelihood of the spiking observations. Those two contributions can be isolated independently by taking two different limits that we will consider in the next two subsections.

## 2.1 Small noise limit

In the limit of small variance of the noise driving the OU process, i.e., $\sigma_W^2 = \epsilon\sigma_{W_0}^2$ with $\epsilon \to 0$, the asymptotic uncertainty $\sigma_\infty^2$ scales with $\epsilon$: $\sigma_\infty^2 = \epsilon\sigma_{W_0}^2/2\theta$ (c.f. Eq. 8 with $\dot{\sigma}^2 = 0$). Then the dynamics of $\mu$ becomes driven only by the prior mean membrane potential $u_r$:

$$\dot{\mu} \simeq -\theta\left(\mu - u_r\right) \tag{10}$$

and so the asymptotic estimated membrane potential will tend to the prior mean membrane potential. This is reasonable since in the small noise limit, the true membrane potential $u_t$ will effectively be very close to $u_r$. Furthermore the convergence time constant of the estimated membrane potential should be matched to the time constant $\theta^{-1}$ of the OU process and this is indeed the case in Eq. 10.

## 2.2 Slow dynamics limit

A second interesting limit is where the time constant of the OU process becomes small, i.e., $\theta = \epsilon\theta_0$ with $\epsilon \to 0$. In this case, the variance of the noise in the OU process must also scale with $\epsilon$, i.e $\sigma_W^2 = \epsilon\sigma_{W_0}^2$, to prevent the process from being unbounded. The variance $\sigma_{OU}^2 = \sigma_{W_0}^2/2\theta_0$ of the OU process is therefore independent of $\epsilon$. In this case, the asymptotic value of the posterior variance becomes $\sigma_\infty^2 = \sqrt{\epsilon}\sigma_{W_0}/\sqrt{\beta\gamma}$ (c.f. Eq. 8 with $\dot{\sigma}^2 = 0$). In the limit of small $\epsilon$, the first term of Eq. 7 scales with $\epsilon$ whereas the second term with $\sqrt{\epsilon}$. We can therefore write:

$$\frac{\sqrt{\gamma}}{\sigma_W}\dot{\mu} \simeq S(t) - \gamma \tag{11}$$

Because the time constant $\theta^{-1}$ of the OU process is slow, the driving force that pulls the membrane potential back to its mean value $u_r$ is weak. Therefore the membrane potential estimation dynamics should rely on the observed spikes rather than on the prior information $u_r$. This is apparent in Eq. 11.

Furthermore, the time constant $\tau = \sqrt{\gamma/\epsilon}/\sigma_{W_0}$ is not fixed but is a function of the mean estimated membrane potential $\mu$. Thus, if the initial estimate $\mu_0 = \mu(0)$ is below the target value $u_r$, $\gamma$

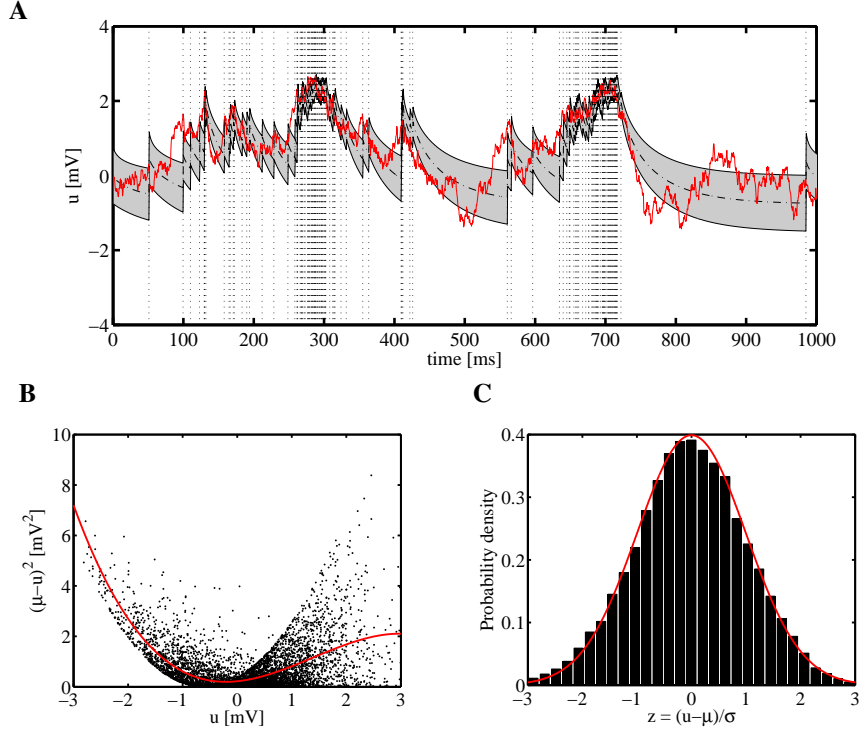

Figure 2: The performance of the optimal on-line estimator. **A.** *Red line:* presynaptic membrane potential, $u$, as a function of time, *vertical dotted lines:* spikes emitted. *Dot-dashed black line:* on-line estimator $\mu$ given by Eq. (7), *gray shading:* $\mu \pm \sigma$, with $\sigma$ given by Eq. (8). **B.** Estimation error $(\mu - u)^2$ as a function of the membrane potential $u$ of the OU process. *Black dots:* estimation error and true membrane potential in individual time steps, *red line:* third order polynomial fit. **C** *Black bars*: histogram of normalized estimation error $z = (\mu - u)/\sigma$. *Red line*: normal distribution $\mathcal{N}(z; 0, 1)$. Parameters were as in Fig. 1, except for $\beta^{-1} = 0.5$ mV .

will be small and hence the time constant $\tau$ will be small as well. As a consequence, each spike will greatly increase the estimate and therefore speed up the approach of this estimate to the true value. As $\mu$ gets closer to the true membrane potential, the time constant increases, leading to an appropriately accurate estimate of the membrane potential. This dynamical time constant therefore helps the estimation avoid the traditional speed vs accuracy trade-off (short time constant are fast but give a noisy estimation; longer time constant are slow but yield a more accurate estimation), by combining the best of the two worlds.

## 3    Depressing synapses as estimators of presynaptic membrane potential

In section 2 we have shown that presynaptic spikes have a varying, context-dependent effect on the optimal on-line estimator of presynaptic membrane potential. In this section we will show that the variability that synaptic depression introduces in postsynaptic responses closely resembles the variability of the optimal estimator.

A simple way to study the similarity between the optimal estimator and short-term plasticity is to consider their steady state filtering properties. As we saw above, according to the optimal estimator, the higher the input firing rate is, the smaller the posterior variance becomes, and therefore the increment due to subsequent spikes should decrease. This is consistent with depressing synapses for which the amount of excitatory postsynaptic current (EPSC) decreases when the stimulation frequency is increased (see Fig. 3).

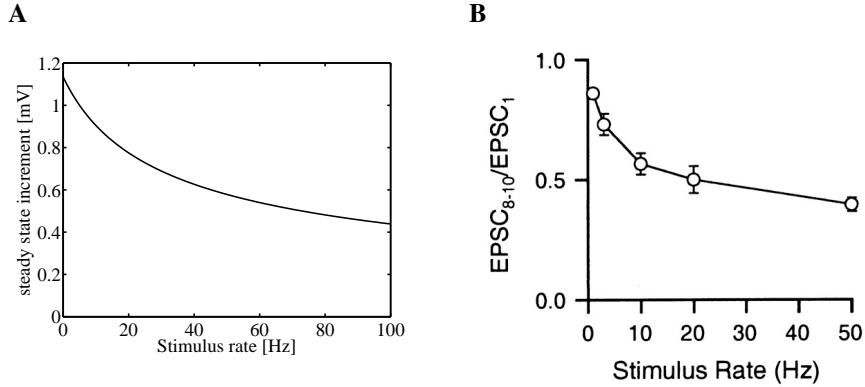

Figure 3: **A.** Steady-state spiking increment $\beta\sigma^2$ of the optimal estimator as a function of $r = \langle S \rangle$ (Eq. 8). **B.** Synaptic depression in the climbing fibre to Purkinje cell synapse: average ($\pm$s.e.m.) normalised "steady-state" magnitude of EPSCs as a function of stimulation frequency. Reproduced from [3].

Importantly, the similarity between the optimal membrane potential estimator and short-term plasticity is not limited to stationary properties. Indeed, the actual dynamics of the optimal estimator (Eqs. 7-9) can be well approximated by the dynamics of synaptic depression. In a canonical model of short-term depression [14], the postsynaptic membrane potential, $v$, changes as

$$\dot{v} = -\frac{v - v_0}{\tau} + J\,Y\,x\,S(t), \qquad \text{with} \qquad \dot{x} = \frac{1 - x}{\tau_D} - Y\,x\,S(t) \qquad (12)$$

where $J$ and $Y$ are constants (synaptic weight and utilisation fraction), and $x$ is a time varying 'resource' variable (e.g. the fraction of presynaptic vesicles ready to fuse to the membrane). Thus, $v$ is increased by each presynaptic spike, and in the absence of spikes it decays to its resting value, $v_0$, with membrane time constant $\tau$. However, the effect of each spike on $v$ is scaled by $x$ which itself is decreased after each spike and increases between spikes back towards one with time constant $\tau_D$. Thus, the postsynaptic potential, $v$, behaves much like the posterior mean of the optimal estimator, $\mu$, while the dynamics of the synaptic resource variable, $x$, closely resemble that of the posterior variance of the optimal estimator, $\sigma^2$. This qualitative similarity can be made more formal under appropriate assumptions, for details see section 3 of supplementary information. Indeed, the capacity of a depressing synapse (with appropriate parameters) to estimate the presynaptic membrane potential can be nearly as good as that of the optimal estimator (Fig. 4, top). Interestingly, although the scaled variance $\sigma^2/\sigma_\infty^2$ does not follow the resource variable dynamics $x$ perfectly just after a spike, these two quantities are virtually identical at the time of the next spike, i.e. when they are used by the membrane potential estimators (Fig. 4, bottom).

## 4    Performance analysis

In order to quantify how well synaptic dynamics with depression perform in estimating presynaptic membrane potentials, we measure performance by the mean-squared error (MSE) between the true membrane potential $u$ and the estimated membrane potential, and compare the MSE of three alternatives estimators.

The simplest model we consider is a static (non-depressing) synapse, in which $v$ is given by Eq. 12 with constant $x = 1$. This estimator has only 3 tuneable parameters: $\tau$, $v_0$ and $J$ ($Y = 1$ is fixed without loss of generality). The second estimator we consider includes synaptic depression, i.e. $x$ is also allowed to vary (Eq. 12). This estimator contains 5 tuneable parameters ($v_0$, $\tau$, $Y$, $J$, $\tau_D$). Finally, we consider the optimal estimator (Eqs. 7-9). This estimator has no tunable parameters. Once the parameters of presynaptic membrane potential dynamics ($\sigma_W$, $\theta$, $u_r$) and spiking ($\beta$, $g_0$) are fixed, the optimal estimator is entirely determined. The comparison of the performance of these three estimators is displayed on Fig. 5. The optimal estimator (black circles) is obviously a lower bound on any type of estimator. For a wide range of parameter values, the depressing synapse performs almost as well as the optimal estimator, and both perform better than the static synapse.

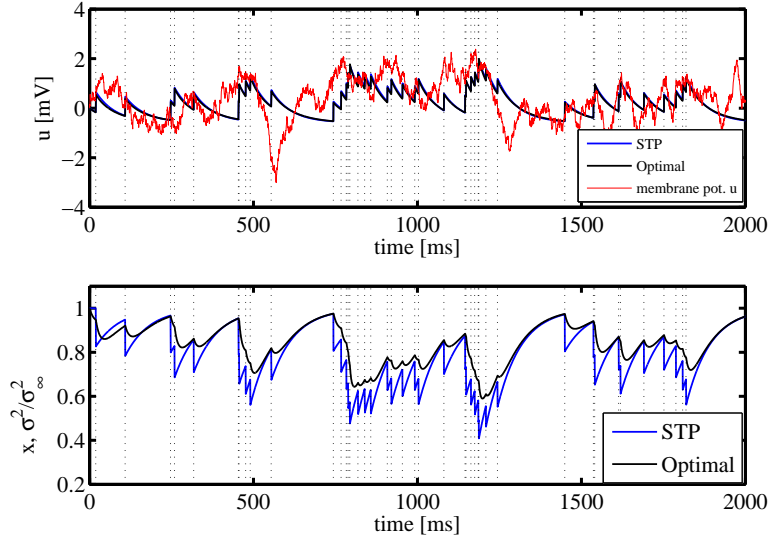

Figure 4: Depressing synapses implement near-optimal estimation of presynaptic membrane potentials. **Top.** *Red line*, and *vertical dotted lines:* membrane potential, $u$, and spikes, $S$, generated by a simulated presynaptic cell (with parameters as in Fig. 1). *Blue line:* postsynaptic potential, $v$, in a depressing synapse (Eq. 12) with all 5 parameters ($J = 4.82$, $\tau = 60.6$ ms, $v_0 = -0.59$ mV, $\tau_d = 64$ ms, Y = 0.17) tuned to minimize the mean squared estimation error, $(u - v)^2$. *Black line:* Posterior mean of the optimal on-line estimator, $\mu$ (Eq. 7). **Bottom.** *Black:* resource variable, $x$, in the depressing synapse (Eq. 12). *Blue:* posterior variance of the optimal estimator, $\sigma^2$ (Eq. 8).

In the slow dynamics limit ($\epsilon \to 0$, see section 2.2), the estimation error of the optimal estimator can even be approximated analytically (see Supplementary Information). In this limit, the error scales with $\sqrt{\sigma_W}$ and therefore scales with $\sqrt[4]{\epsilon}$. As can be seen on Fig. 5B, for small $\epsilon$, the analytical expression is consistent with the simulations.

## 5  Discussion

Synapses are a cornerstone of computation in networks, and are highly complex dynamical systems involving more than a thousand different types of protein. One prominent feature of their dynamics is significant short-term changes in efficacy; these belie the sort of single fixed, or slowly changing, weights popular in most neural models. We interpreted short-term synaptic depression, a key feature of synaptic dynamics, as solving the fundamental computational task of estimating the analog membrane potential of the presynaptic cell from observed spikes. Steady-state and dynamical properties of a Bayes-optimal estimator are well-matched by a canonical model of depression; using a fixed synaptic efficacy instead leads to a highly suboptimal estimator.

Our theory is readily testable, since it suggests a precise relationship between quantities that have been subject to extensive, separate, empirical study — namely the statistics of a neuron's membrane potential dynamics (captured by the parameters of Eq. (2)), the form of its spiking non-linearity (described by Eq. (5)), and the synaptic depression it expresses in its efferent synapses. Accounting for the observation that different efferent synapses of the same cell can express different forms of short-term synaptic plasticity [15] remains a challenge; one obvious possibility is that different synapses are estimating different aspects or functions of the membrane potential.

Our approach is almost dual to that explored in [16]. For that model, the spike generation mechanism of the presynaptic neuron was modified such that even a simple read-out mechanism with fixed efficacies could correctly decode the analogue quantity encoded presynaptically. By contrast, we considered a standard model of spiking [17], and thereby derived an explanation for the evident fact that synapses are not in fact fixed.

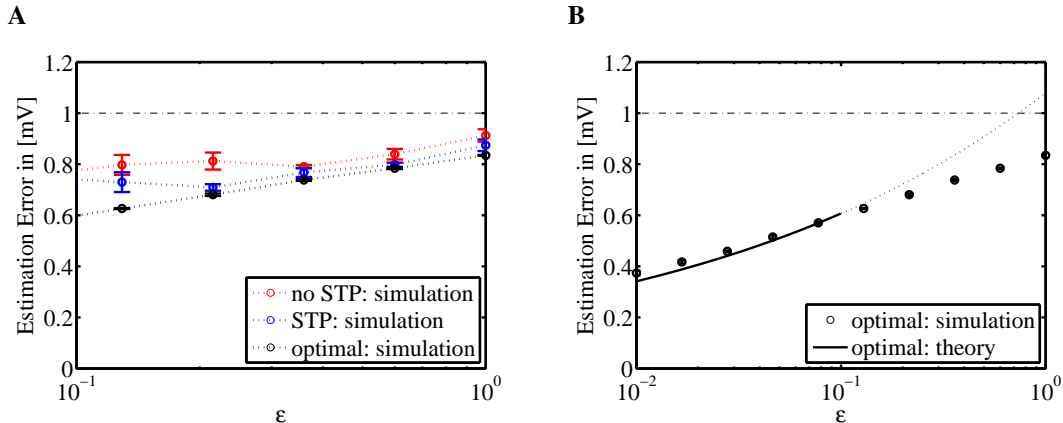

Figure 5: **A.** Comparing the estimation error for different membrane potential estimators as a function of $\epsilon$. ($\theta = \epsilon \theta_0$, $\sigma_{\mathrm{W}}^2 = \epsilon \sigma_{W_0}^2$). *Black:* asymptotic error of the optimal estimator. *Blue:* depressing synapse with its 5 tuneable parameters (see text) being optimised for each value of $\epsilon$. *Red:* static synapse with its 3 tuneable parameters (see text) being optimised. Total simulated time was 5 min. *Horizontal dot-dashed line*: upper bound on the estimation error given by $\sigma_{\mathrm{OU}} = \sigma_{\mathrm{W}}/\sqrt{2\theta} = 1$. **B.** Analysing the estimation error of the optimal estimator in the slow dynamics limit ($\epsilon \to 0$). *Solid line:* analytical approximation (Eq. 31 in the Supplementary Information), *circles:* simulation, *horizontal dot-dashed line:* as in *A*.

There are several avenues to extend the present analysis. For example, it would be important to understand in more quantitative detail the mapping between the parameters of the process generating the presynaptic membrane potential and spikes, and the parameters of synaptic depression that will best realize the corresponding optimal estimator. We present some preliminary derivations in the supplementary material that seem to yield at least the right ball-park values for optimal synaptic dynamics. This should also enable us to explore the particular parameter regimes in which depressing synapses have the most (or least) advantage over static synapses in terms of estimation performance, as in Fig. 5. We should also consider a meta-plasticity rule that suitably adapts the parameters of the short-term dynamics in the light of the statistics of spiking.

Our assumption about the prior distribution of presynaptic membrane potential dynamics is highly restrictive. A broader scheme that has previously been explored is that it follow a Gaussian process model [18, 19] with a more general covariance function. Recursive estimation is often a reasonable approximation in such cases, even for those covariance functions, for instance enforcing smoothness, for which it cannot be exact. One interesting property of smooth trajectories is that a couple of spikes arriving in quick succession may be diagnostic of an upward-going trend in membrane potential which is best decoded with increasing, i.e., facilitating, rather than decreasing, postsynaptic responses. Thus it may be possible to encompass other forms of short term plasticity within our scheme.

The spike generation process can also be extended to incorporate refractoriness, bursting, and other forms of non-Poisson behaviour, eg. as in [20]. Similarly, synaptic failures could also be considered. We hope through our theory to be able to provide a teleological account of the rich complexities of real synaptic inconstancy.

## Acknowledgements

Funding was from the Gatsby Charitable Foundation (PD) and the Wellcome Trust (JPP, ML and PD).

## Footnotes

[1]Note that the exponential gain function is a convenient choice since the product of a Gaussian and an exponential gives again an (unnormalised) Gaussian (see Supplementary Information). Furthermore, the exponential gain function has also some experimental support [13].

# References

[1] Abbott, L.F. & Regehr, W.G. Synaptic computation. *Nature* **431**, 796–803 (2004).

[2] Zucker, R. & Regehr, W. Short-term synaptic plasticity. *Annual Review of Physiology* **64**, 355–405 (2002).

[3] Dittman, J., Kreitzer, A. & Regehr, W. Interplay between facilitation, depression, and residual calcium at three presynaptic terminals. *Journal of Neuroscience* **20**, 1374 (2000).

[4] Abbott, L.F., Varela, J.A., Sen, K. & Nelson, S.B. Synaptic depression and cortical gain control. *Science* **275**, 220–224 (1997).

[5] Cook, D., Schwindt, P., Grande, L. & Spain, W. Synaptic depression in the localization of sound. *Nature* **421**, 66–70 (2003).

[6] Goldman, M., Maldonado, P. & Abbott, L. Redundancy reduction and sustained firing with stochastic depressing synapses. *Journal of Neuroscience* **22**, 584 (2002).

[7] Ermentrout, B. Neural networks as spatio-temporal pattern-forming systems. *Reports on Progress in Physics* **61**, 353 (1998).

[8] Shu, Y., Hasenstaub, A., Duque, A., Yu, Y. & McCormick, D. Modulation of intracortical synaptic potentials by presynaptic somatic membrane potential. *Nature* **441**, 761–765 (2006).

[9] Eden, U., Frank, L., Barbieri, R., Solo, V. & Brown, E. Dynamic analysis of neural encoding by point process adaptive filtering. *Neural Computation* **16**, 971–998 (2004).

[10] Bobrowski, O., Meir, R. & Eldar, Y. Bayesian filtering in spiking neural networks: Noise, adaptation, and multisensory integration. *Neural Computation* **21**, 1277–1320 (2009).

[11] Dayan, P. & Abbott, L.F. *Theoretical Neuroscience* (MIT Press, Cambridge, 2001).

[12] Markram, H. & Tsodyks, M. Redistribution of synaptic efficacy between neocortical pyramidal neurons. *Nature* **382**, 807–810 (1996).

[13] Jolivet, R., Rauch, A., Lüscher, H.R. & Gerstner, W. Predicting spike timing of neocortical pyramidal neurons by simple threshold models. *J. Computational Neuroscience* **21**, 35–49 (2006).

[14] Mongillo, G., Barak, O. & Tsodyks, M. Synaptic theory of working memory. *Science* **319**, 1543 (2008).

[15] Markram, H., Wu, Y. & Tosdyks, M. Differential signaling via the same axon of neocortical pyramidal neurons. *Proc. Natl. Acad. Sci. USA* **95**, 5323–5328 (1998).

[16] Deneve, S. Bayesian spiking neurons I: inference. *Neural Computation* **20**, 91–117 (2008).

[17] Gerstner, W. & Kistler, W.K. *Spiking Neuron Models* (Cambridge University Press, Cambridge UK, 2002).

[18] Cunningham, J., Yu, B., Shenoy, K. & Sahani, M. Inferring neural firing rates from spike trains using Gaussian processes. *Advances in Neural Information Processing Systems* **20**, 329–336 (2008).

[19] Huys, Q., Zemel, R., Natarajan, R. & Dayan, P. Fast population coding. *Neural Computation* **19**, 404–441 (2007).

[20] Pillow, J. *et al.* Spatio-temporal correlations and visual signalling in a complete neuronal population. *Nature* **454**, 995–999 (2008).

